# Metamorphosis Networks:
# An Alternative to Constructive Methods

**Brian V. Bonnlander**      **Michael C. Mozer**
Department of Computer Science &
Institute of Cognitive Science
University of Colorado
Boulder, CO 80309-0430

## Abstract

Given a set of training examples, determining the appropriate number of free parameters is a challenging problem. Constructive learning algorithms attempt to solve this problem automatically by adding hidden units, and therefore free parameters, during learning. We explore an alternative class of algorithms—called *metamorphosis algorithms*—in which the number of units is fixed, but the number of free parameters gradually increases during learning. The architecture we investigate is composed of RBF units on a lattice, which imposes flexible constraints on the parameters of the network. Virtues of this approach include variable subset selection, robust parameter selection, multiresolution processing, and interpolation of sparse training data.

## 1  INTRODUCTION

Generalization performance on a fixed-size training set is closely related to the number of free parameters in a network. Selecting either too many or too few parameters can lead to poor generalization. Geman et al. (1991) refer to this problem as the *bias/variance dilemma*: introducing too many free parameters incurs high variance in the set of possible solutions, and restricting the network to too few free parameters incurs high bias in the set of possible solutions.

Constructive learning algorithms (e.g., Fahlman & Lebiere, 1990; Platt, 1991) have

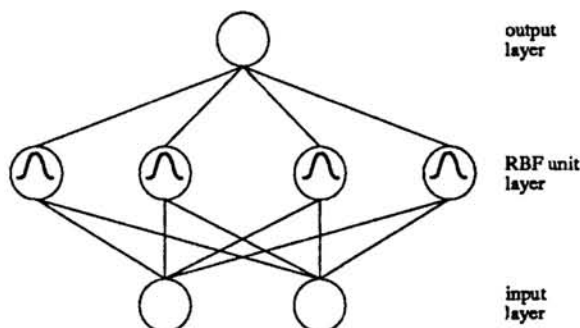

Figure 1: Architecture of an RBF network.

been proposed as a way of automatically selecting the number of free parameters in the network during learning. In these approaches, the learning algorithm gradually increases the number of free parameters by adding hidden units to the network. The algorithm stops adding hidden units when some validation criterion indicates that network performance is good enough.

We explore an alternative class of algorithms—called *metamorphosis algorithms*— for which the number of units is fixed, but heavy initial constraints are placed on the unit response properties. During learning, the constraints are gradually relaxed, increasing the flexibility of the network. Within this general framework, we develop a learning algorithm that builds the virtues of recursive partitioning strategies (Breiman et al., 1984; Friedman, 1991) into a Radial Basis Function (RBF) network architecture. We argue that this framework offers two primary advantages over constructive RBF networks: for problems with low input variable interaction, it can find solutions with far fewer free parameters, and it is less susceptible to noise in the training data. Other virtues include multiresolution processing and built-in interpolation of sparse training data.

Section 2 introduces notation for RBF networks and reviews the advantages of using these networks in constructive learning. Section 3 describes the idea behind metamorphosis algorithms and how they can be combined with RBF networks. Section 4 describes the advantages of this class of algorithm. The final section suggests directions for further research.

## 2    RBF NETWORKS

RBF networks have been used successfully for learning difficult input-output mappings such as phoneme recognition (Wettschereck & Dietterich, 1991), digit classification (Nowlan, 1990), and time series prediction (Moody & Darken, 1989; Platt, 1991). The basic architecture is shown in Figure 1. The response properties of each RBF unit are determined by a set of parameter values, which we'll call a *pset*. The pset for unit $i$, denoted $\mathbf{r}_i$, includes: the center location of the RBF unit in the input space, $\mu_i^r$; the width of the unit, $\sigma_i^r$; and the strength of the connection(s) from the RBF unit to the output unit(s), $\mathbf{h}_i^r$.

One reason why RBF networks work well with constructive algorithms is because

the hidden units have the property of *noninterference*: the nature of their activation functions, typically Gaussian, allows new RBF units to be added without changing the global input-output mapping already learned by the network.

However, the advantages of constructive learning with RBF networks diminish for problems with high-dimensional input spaces (Hartman & Keeler, 1991). For these problems, a large number of RBF units are needed to cover the input space, even when the number of input dimensions relevant for the problem is small. The relevant input dimensions can be different for different parts of the input space, which limits the usefulness of a global estimation of input dimension relevance, as in Poggio and Girosi (1990). Metamorphosis algorithms, on the other hand, allow RBF networks to solve problems such as these without introducing a large number of free parameters.

# 3   METAMORPHOSIS ALGORITHMS

Metamorphosis networks contrast with constructive learning algorithms in that the number of units in the network remains fixed, but degrees of freedom are gradually added during learning. While metamorphosis networks have not been explored in the context of supervised learning, there is at least one instance of a metamorphosis network in unsupervised learning: a Kohonen net. Units in a Kohonen net are arranged on a lattice; updating the weights of a unit causes weight updates of the unit's neighbors. Units nearby on the lattice are thereby forced to have similar responses, reducing the effective number of free parameters in the network. In one variant of Kohonen net learning, the neighborhood of each unit gradually shrinks, increasing the degrees of freedom in the network.

## 3.1   MRBF NETWORKS

We have applied the concept of metamorphosis algorithms to ordinary RBF networks in supervised learning, yielding *MRBF networks*. Units are arranged on an $n$-dimensional lattice, where $n$ is picked ahead of time and is unrelated to the dimensionality of the input space. The response of RBF unit $i$ is constrained by deriving its pset, $\mathbf{r}_i$, from a collection of *underlying* psets, each denoted $\mathbf{u}_j$, that also reside on the lattice. The elements of $\mathbf{u}_j$ correspond to those of $\mathbf{r}_i$: $\mathbf{u}_j = (\mu_j^{\mathbf{u}}, \sigma_j^{\mathbf{u}}, \mathbf{h}_j^{\mathbf{u}})$. Due to the orderly arrangement of the $\mathbf{u}_j$, the lattice is divided into nonoverlapping hyperrectangular regions that are bounded by $2^n$ $\mathbf{u}_j$. Consequently, each $\mathbf{r}_i$ is *enclosed* by $2^n$ $\mathbf{u}_j$. The pset $\mathbf{r}_i$ can then be derived by linear interpolation of the enclosing underlying psets $\mathbf{u}_j$, as shown in Figure 2 for a one-dimensional lattice.

Learning in MRBF networks proceeds by minimizing an error function E in the $\mathbf{u}_j$ components via gradient descent:

$$\Delta \mu_{jk}^{\mathbf{u}} = -\eta \sum_{i \in \text{NEIGH}_j} \frac{\partial \text{E}}{\partial \mu_{ik}^{\mathbf{r}}} \frac{\partial \mu_{ik}^{\mathbf{r}}}{\partial \mu_{jk}^{\mathbf{u}}}$$

where $\text{NEIGH}_j$ is the set of RBF units whose values are affected by underlying pset $j$, and $k$ indexes the input units of the network. The update expression is similar for $\sigma_j^{\mathbf{u}}$ and $\mathbf{h}_j^{\mathbf{u}}$. To better condition the search space, instead of optimizing the

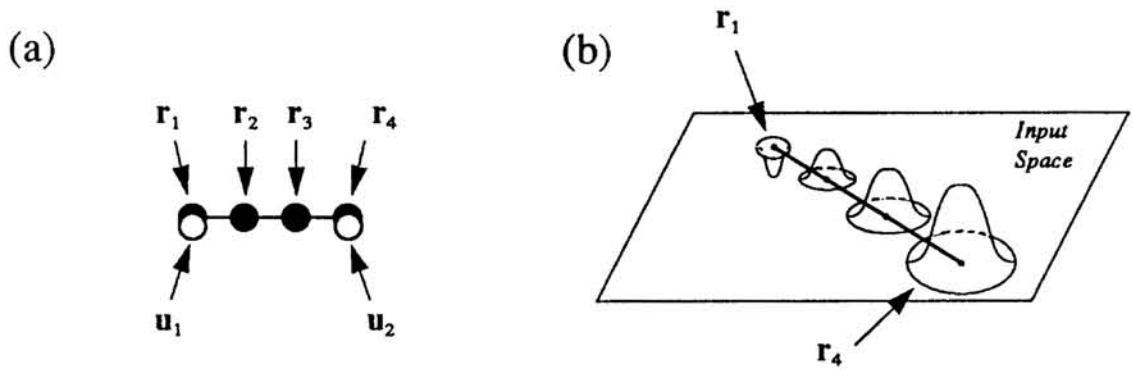

Figure 2: Constrained RBF units. (a) Four RBF units with psets $r_1$–$r_4$ are arranged on a one-dimensional lattice, enclosed by underlying psets $u_1$ and $u_2$. (b) An input space representation of the constrained RBF units. RBF center locations, widths, and heights are linearly interpolated.

$\sigma_i^r$ directly, we follow Nowlan and Hinton's (1991) suggestion of computing each RBF unit width according to the transformation $\sigma_i^r = exp(\gamma_i/2)$ and searching for the optimum value of $\gamma_i$. This forces RBF widths to remain positive and makes it difficult for a width to approach zero.

When a local optimum is reached, either learning is stopped or additional underlying psets are placed on the lattice in a process called *metamorphosis*.

## 3.2   METAMORPHOSIS

Metamorphosis is the process that gradually adds new degrees of freedom to the network during learning. For the MRBF network explored in this paper, introducing new free parameters corresponds to placing additional underlying psets on the lattice. The new psets split one hyperrectangular region—an $n$-dimensional sublattice bounded by $2^n$ underlying psets—into two nonoverlapping hyperrectangular regions. To achieve this, $2^{n-1}$ additional underlying psets, which we call the *split group*, are required (Figure 3). The splitting process implements a *recursive partitioning* strategy similar to the strategies employed in the CART (Breiman et al., 1984) and MARS (Friedman, 1991) statistical learning algorithms.

Many possible rules for region splitting exist. In the simulations presented later, we consider every possible region and every possible split of the region into two subregions. For each split group $k$, we compute the *tension* of the split, defined as

$$\sum_{j \in \text{split group } k} \left\| \frac{\partial E}{\partial u_j} \right\|^2.$$

We then select the split group that has the greatest tension. This heuristic is based on the assumption that the error gradient at the point in weight space where a split would take place reflects the long-term benefit of that split.

It may appear that this splitting process is computationally expensive, but it can be implemented quite efficiently; the cost of computing all possible splits and choosing

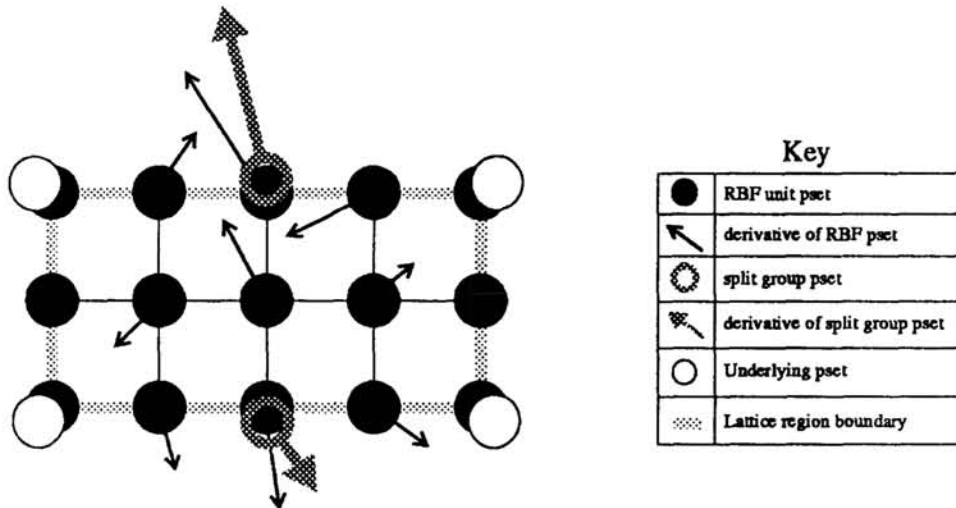

| Key | |
|---|---|
| ● | RBF unit pset |
| ↖ | derivative of RBF pset |
| ◌ | split group pset |
| ↘ | derivative of split group pset |
| ○ | Underlying pset |
| ⠿ | Lattice region boundary |

Figure 3: Computing the tension of a split group. Arrows are meant to represent derivatives of corresponding pset components.

the best one is linear in the number of RBF units on the lattice.

# 4     VIRTUES OF METAMORPHOSIS NETS

## 4.1     VARIABLE SUBSET SELECTION

One advantage of MRBF networks is that they can perform *variable subset selection*; that is, they can select a subset of input dimensions more relevant to the problem and ignore the other input dimensions. This is also a property of other recursive partitioning algorithms such as CART and MARS. In MRBF networks, however, region splitting occurs on a lattice structure, rather than in the input space. Consequently, the learning algorithm can orient a small number of regions to fit data that is not aligned with the lattice to begin with. CART and MARS have to create many regions to fit this kind of data (Friedman, 1991).

To see if this style of learning algorithm could learn to solve a difficult problem, we trained an MRBF network on the Mackey-Glass chaotic time series. Figure 4(a) compares normalized RMS error on the test set with Platt's (1991) RAN algorithm as the number of parameters increases during learning. Although RAN eventually finds a superior solution, the MRBF network requires a much smaller number of free parameters to find a reasonably accurate solution. This result agrees with the idea that ordinary RBF networks must use many free parameters to cover an input space with RBF units, whereas MRBF networks may use far fewer by concentrating resources on only the most relevant input dimensions.

## 4.2     ROBUST PARAMETER SELECTION

In RBF networks, the local response of a hidden unit makes it difficult for back propagation to move RBF centers far from where they are originally placed. Consequently, the choice of initial RBF center locations is critical for constructive al-

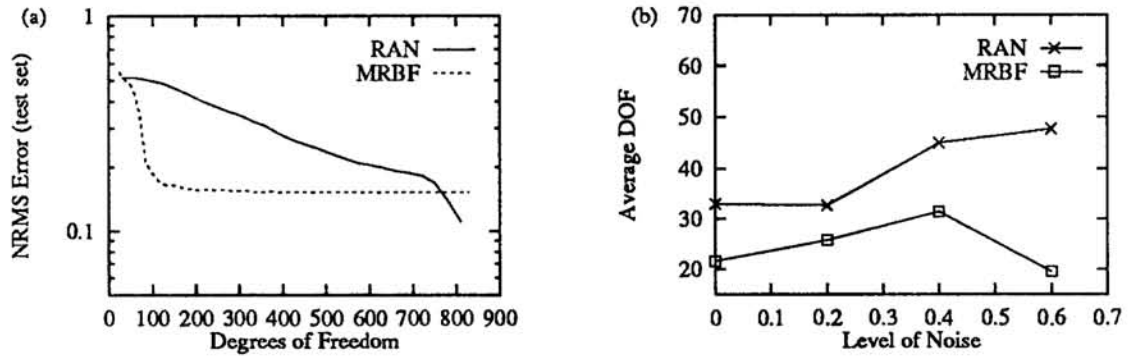

Figure 4: (a) Comparison on the Mackey-Glass chaotic time series. The curves for RAN and MRBF represent an average over ten and three simulation runs, respectively. The simulations used 300 training patterns and 500 test patterns as described in (Platt 1991). Simulation parameters for RAN match those reported in (Platt 1991) with $\epsilon = 0.02$. (b) Gaussian noise was added to the function $y = sin8\pi x$, $0 < x < 1$, where the task was to predict y given x. The horizontal axis represents the standard deviation of the Gaussian distribution. For both algorithms, 20 simulations were run at each noise level. The number of degrees of freedom (DOF) needed to achieve a fixed error level was averaged.

gorithms. Poor choices could result in the allocation of more RBF units than are necessary. One apparent weakness of the RAN algorithm is that it chooses RBF center locations based on individual examples, which makes it susceptible to noise. Metamorphosis in MRBF networks, on the other hand, is based on the more global measure of tension.

Figure 4(b) shows the average number of degrees of freedom allocated by RAN and an MRBF network on a simple, one-dimensional function approximation task. Gaussian noise was added to the target output values in the training and test sets. As the amount of noise increases, the average number of free parameters allocated by RAN also increases, whereas for the MRBF network, the average remains low.

One interesting property of RAN is that allocating many extra RBF units does not necessarily hurt generalization performance. This is true when RAN starts with wide RBF units and decreases the widths of candidate RBF units slowly. The main disadvantage to this approach is wasted computational resources.

### 4.3   MULTIRESOLUTION PROCESSING

Our approach has the property of initially finding solutions sensitive to coarse problem features and using these solutions to find refinements more sensitive to finer features (Figure 5). This idea of *multiresolution processing* has been studied in the context of computer vision relaxation algorithms and is a property of algorithms proposed by other authors (e.g. Moody, 1989, Platt, 1991).

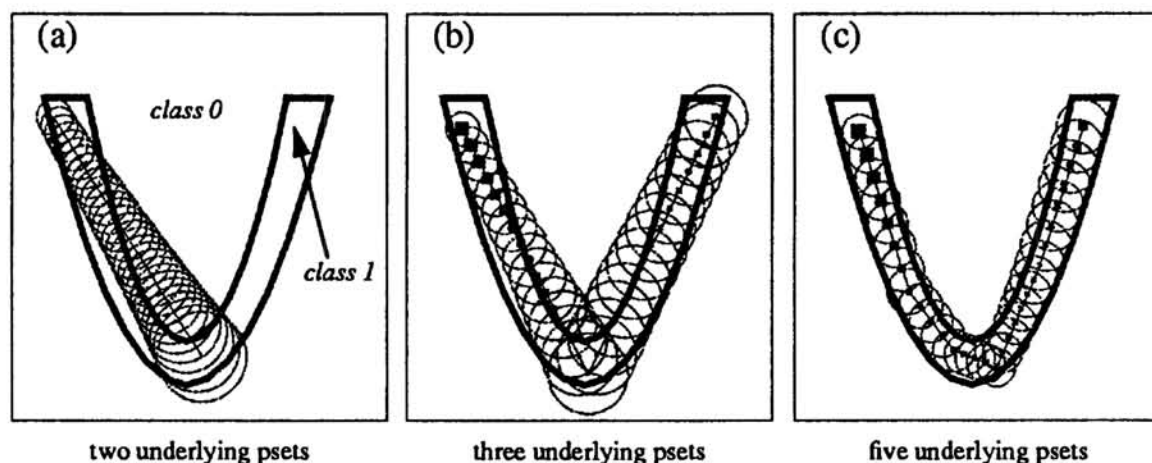

two underlying psets      three underlying psets      five underlying psets

Figure 5: Example of multiresolution processing. The figure shows performance on a two-dimensional classification task, where the goal is to classify all inputs inside the U-shape as belonging to the same category. An MRBF network is constrained using a one-dimensional lattice. Circles represent RBF widths, and squares represent the height of each RBF.

## 4.4  INTERPOLATION OF SPARSE TRAINING DATA

For a problem with sparse training data, it is often necessary to make assumptions about the appropriate response at points in the input space far away from the training data. Like nearest-neighbor algorithms, MRBF networks have such an assumption built in. The constrained RBF units in the network serve to *interpolate* the values of underlying psets (Figure 6). Although ordinary RBF networks can, in principle, interpolate between sparse data points, the local response of an RBF unit makes it difficult to find this sort of solution by back propagation.

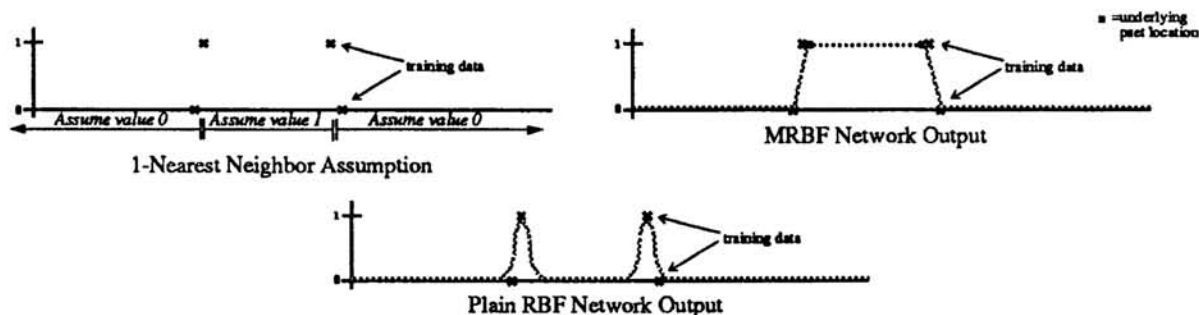

Figure 6: Assumptions made for sparse training data on a task with a one-dimensional input space and one-dimensional output space. Target output values are marked with an 'x'. Like nearest-neighbor algorithms, the assumption made by MRBF networks causes network response to interpolate between sparse data points. This assumption is not built into ordinary RBF networks.

## 5   DIRECTIONS FOR FURTHER RESEARCH

In our simulations to date, we have not observed astonishingly better generalization performance with metamorphosis nets than with alternative approaches, such as Platt's RAN algorithm. Nonetheless, we believe the approach worthy of further exploration. We've examined but one type of metamorphosis net and in only a few domains. The sorts of investigations we are considering next include: substituting finite-element basis functions for RBFs, implementing a "soft" version of the RBF pset constraint using regularization techniques, and using a supervised learning algorithm similar to Kohonen networks, where updating the weights of a unit causes weight updates of the unit's neighbors.

### Acknowledgements

This research was supported by NSF PYI award IRI–9058450 and grant 90–21 from the James S. McDonnell Foundation. We thank John Platt for providing the Mackey-Glass time series data, and Chris Williams, Paul Smolensky, and the members of the Boulder Connectionist Research Group for helpful discussions.

### References

L. Breiman, J. Friedman, R. A. Olsen & C. J. Stone. (1984) *Classification and Regression Trees.* Belmont, CA: Wadsworth.

S. E. Fahlman & C. Lebiere. (1990) The cascade-correlation learning architecture. In D. S. Touretzky (ed.), *Advances in Neural Information Processing Systems 2*, 524-532. San Mateo, CA: Morgan Kaufmann.

J. Friedman. (1991) Multivariate Adaptive Regression Splines. *Annals of Statistics* **19**:1-141.

S. Geman, E. Bienenstock & R. Doursat. (1992) Neural networks and the bias/variance dilemma. *Neural Computation* **4**(1):1-58.

E. Hartman & J. D. Keeler. (1991) Predicting the future: advantages of semilocal units. *Neural Computation* **3**(4):566-578.

T. Kohonen. (1982) Self-organized formation of topologically correct feature maps. *Biological Cybernetics* **43**:59-69.

J. Moody & C. Darken. (1989) Fast learning in networks of locally-tuned processing units. *Neural Computation* **1**(2):281-294.

J. Moody. (1989) Fast learning in multi-resolution hierarchies. In D. S. Touretzky (ed.), *Advances in Neural Information Processing 1*, 29-39. San Mateo, CA: Morgan Kaufmann.

S. J. Nowlan. (1990) Maximum likelihood competition in RBF networks. Tech. Rep. CRG-TR-90-2, Department of Computer Science, University of Toronto, Toronto, Canada.

S. J. Nowlan & G. Hinton. (1991) Adaptive soft weight-tying using Gaussian Mixtures. In Moody, Hanson, & Lippmann (eds.), *Advances in Neural Information Processing 4*, 993-1000. San Mateo, CA: Morgan-Kaufmann.

J. Platt. (1991) A resource-allocating network for function interpolation. *Neural Computation* **3**(2):213-225.

T. Poggio & F. Girosi. (1990) Regularization algorithms for learning that are equivalent to multilayer networks. *Science* **247**:978-982.

D. Wettschereck & T. Dietterich. (1991) Improving the performance of radial basis function networks by learning center locations. In Moody, Hanson, & Lippmann (eds.), *Advances in Neural Info. Processing 4*, 1133-1140. San Mateo, CA: Morgan Kaufmann.